# A productive, systematic framework for the representation of visual structure

**Shimon Edelman**
232 Uris Hall, Dept. of Psychology
Cornell University
Ithaca, NY 14853-7601
*se37@cornell.edu*

**Nathan Intrator**
Institute for Brain and Neural Systems
Box 1843, Brown University
Providence, RI 02912
*Nathan_Intrator@brown.edu*

## Abstract

We describe a unified framework for the understanding of structure representation in primate vision. A model derived from this framework is shown to be effectively systematic in that it has the ability to interpret and associate together objects that are related through a rearrangement of common "middle-scale" parts, represented as image fragments. The model addresses the same concerns as previous work on compositional representation through the use of *what+where* receptive fields and attentional gain modulation. It does not require prior exposure to the individual parts, and avoids the need for abstract symbolic binding.

## 1  The problem of structure representation

The focus of theoretical discussion in visual object processing has recently started to shift from problems of recognition and categorization to the representation of object structure. Although view- or appearance-based solutions for these problems proved effective on a variety of object classes [1], the "holistic" nature of this approach – the lack of explicit representation of relational structure – limits its appeal as a general framework for visual representation [2].

The main challenges in the processing of structure are productivity and systematicity, two traits commonly attributed to human cognition. A visual system is productive if it is open-ended, that is, if it can deal effectively with a potentially infinite set of objects. A visual representation is systematic if a well-defined change in the spatial configuration of the object (e.g., swapping top and bottom parts) causes a principled change in the representation (e.g., the interchange of the representations of top and bottom parts [3, 2]). A solution commonly offered to the twin problems of productivity and systematicity is compositional representation, in which symbols standing for generic parts drawn from a small repertoire are bound together by categorical symbolically coded relations [4].

## 2 The Chorus of Fragments

In visual representation, the need for symbolic binding may be alleviated by using location in the visual field in lieu of the abstract frame that encodes object structure. Intuitively, the constituents of the object are then bound to each other by virtue of residing in their proper places in the visual field; this can be thought of as a pegboard, whose spatial structure supports the arrangement of parts suspended from its pegs. This scheme exhibits shallow compositionality, which can be enhanced by allowing the "pegboard" mechanism to operate at different spatial scales, yielding effective systematicity across levels of resolution. Coarse coding the constituents (e.g., representing each object fragment in terms of its similarities to some basis shapes) will render the scheme productive. We call this approach to the representation of structure the Chorus of Fragments (CoF; [5]).

### 2.1 Neurobiological building blocks

*What+Where cells.* The representation of spatially anchored object fragments postulated by the CoF model can be supported by *what+where* neurons, each tuned both to a certain shape class *and* to a certain range of locations in the visual field. Such cells have been found in the monkey in areas V4 and posterior IT [6], and in the prefrontal cortex [7].

*Attentional gain fields.* To decouple the representation of object structure from its location in the visual field, one needs a version of the *what+where* mechanism in which the response of the cell depends not merely on the location of the stimulus with respect to fixation (as in classical receptive fields), but also on its location with respect to the focus of attention. Indeed, modulatory effects of object-centered attention on classical RF structure (*gain fields*) have been found in area V4 [8].

### 2.2 Implemented model

Our implementation of the CoF model involves *what+where* cells with attention-modulated gain fields, and is aimed at productive and systematic treatment of composite shapes in object-centered coordinates. It operates directly on gray-level images, pre-processed by a model of the primary visual cortex [9], with complex-cell responses modified to use the MAX operation suggested in [10]. In the model, one *what+where* unit is assigned to the top and one to the bottom fragment of the visual field, each extracted by an appropriately configured Gaussian gain profile (Figure 2, left). The units are trained (1) to discriminate among five objects, (2) to tolerate translation within the hemifield, and (3) to provide an estimate of the reliability of its output, through an autoassociation mechanism attempting to reconstruct the stimulus image [11, 12]. Within each hemifield, the five outputs of a unit can provide a coarse coding of novel objects belonging to the familiar category, in a manner useful for translation-tolerant recognition [13]. The reliability estimate carries information about category, allowing outputs for objects from other categories to be squelched. Most importantly, due to the spatial localization of the unit's receptive field, the system can distinguish between different configurations of the same shapes, while noting the fragment-wise similarities.

We assume that during learning the system performs multiple fixations of the target object, effectively providing the *what+where* units with a basis for spanning the

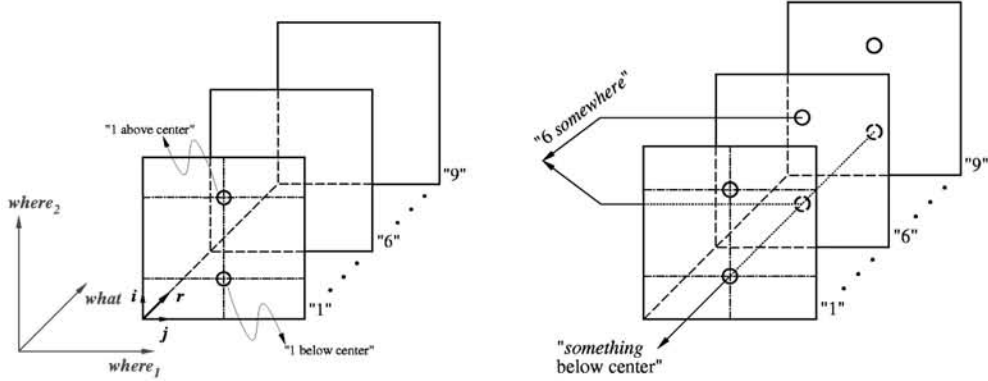

Figure 1: *Left:* the CoF model conceptualized as a "computation cube" trained to distinguish among three fragments (1, 6, 9), each possibly appearing at two locations (above or below the center of attention). A parallel may be drawn between the computation cube and a cortical hypercolumn; in the inferotemporal cortex, cells selective for specific shapes may be arranged in columns, with the dimension perpendicular to the cortical surface encoding different variants of the same shape [14]. It is not known whether the attention-centered location of the shape, which affects the responses of V4 cells [8], is mapped in an orderly fashion onto some physical dimension(s) of the cortex. *Right:* the estimation of the marginal probabilities of shapes, which can be used to decide whether to allocate a unit coding for their composition, can be carried out simply by summing the activities of units along the different dimensions of the computation cube.

space of stimulus translations. It is up to the model, however, to figure out that the objects may be composed of recurring fragments, and to self-organize in a manner that would allow it to deal with novel configurations of those fragments. This problem, which arises both at the level of fragments and of their constituent features, can be addressed within the Minimum Description Length (MDL) framework.

Specifically, we propose to construct receptive fields (RFs) for composite objects so as to capture the deviation from independence between the probability distributions of the responses of RFs tuned to their fragments. This implies a savings in the description length of the composite object. Suppose, for example, that $rf_1$ is the response of a unit tuned roughly to the top half of the character 6 and $rf_2$ – the response of a unit tuned to its bottom half. The construction of a more complex RF combining the responses of these two units will be justified when

$$P(rf_1, rf_2) \gg P(rf_1)P(rf_2) \qquad (1)$$

or, more practically, when some measure of deviation from independence between $P(rf_1)$ and $P(rf_2)$ is large (the simplest such measure would be the covariance, namely, the second moment of the joint distribution but we believe that higher moments may also be required, as suggested by the extensive work on measuring deviation from Gaussian distributions).

By this criterion, a composite RF will be constructed that recognizes the two "parts"

of the character 6 when they are appropriately located: the probability on the LHS of eq. 1 in that case would be proportional to 1/10, while the probability of the RHS would be proportional to 1/100 (assuming that all characters are equiprobable, and that their fragments never appear in isolation). At the same time, a composite RF tuned, say, to 6 above 3 (see section 3) will not be allocated, because the probability of such a complex feature as measured by either the RHS or the LHS of eq. 1 is proportional to 1/100. We note that this feature analysis can be performed on the marginal probabilities of the corresponding fragments, which are by definition less sensitive to image parameters such as the exact location or scale, and can be based on a family of features (cf. Figure 1). A discussion of this approach and of its relationship to the reconstruction constraint we impose when training the fragment-tuned modules is beyond the scope of this paper.

A parallel can be drawn between the MDL framework just outlined and the findings concerning *what+where* cells and gain fields in the shape processing pathway in the monkey cortex. Under the interpretation we propose, the features at all levels of the hierarchy are coarsely coded, and each feature is associated with a rough location in the visual field, so that composite features necessarily represent more complex spatial structure than their constituents, without separately implemented binding, and without a combinatorial proliferation of features. The computational experiments described below concentrate on these novel characteristics of our model, rather than on the standard MDL machinery.

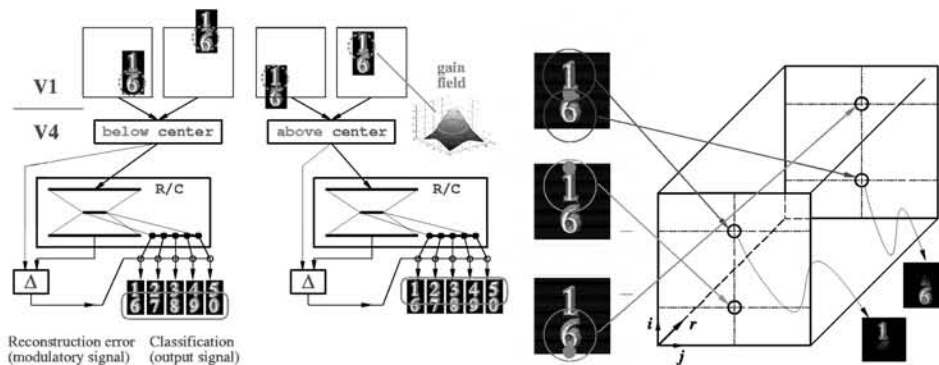

Figure 2: The CoF model, trained on five composite objects (1 over 6, 2 over 7, etc.). *Left:* the model consists of two *what+where* units, responsible for the bottom and the top fragments of the stimulus, respectively. Gain fields (boxes labeled `below center` and `above center`) steer each input fragment to the appropriate unit. The learning mechanism (`R/C`, for Reconstruction/Classification) was implemented as a radial basis function network. The reconstruction error ($\Delta$) modulates the classification outputs. *Right:* training the model, viewed as a computation cube. Multiple fixations of the stimulus (of which three are illustrated), along with Gaussian windows selecting stimulus fragments, allow the system to learn *what+where* responses. A cell would only be allocated to a given fragment if it recurs in the company of a variety of other fragments, as warranted by the ratio between their joint probability and the product of the corresponding marginal probabilities (cf. eq. 1 and Figure 1, right; this criterion has not yet been incorporated into the CoF training scheme).

# 3 Computational experiments

We conducted three experiments that examined the properties of the structured representations emerging from the CoF model. The first experiment (reported elsewhere [13]), involved animal-like shapes and aimed at demonstrating basic productivity and systematicity. We found that the CoF model is capable of systematically interpreting composite objects to which it was not previously exposed (for example, a half-goat and half-lion chimera is represented as such, by an ensemble of units trained to discriminate between three altogether different animals).

In the second experiment, a version of the CoF model (Figure 2) was charged with learning to reuse fragments of the members of the training set — five bipartite objects composed of shapes of numerals from 1 through 0 — in interpreting novel compositions of the same fragments. The gain field mechanism built into the CoF model allowed it to respond largely systematically to the learned fragments even when these were shown in novel locations, both absolute, and relative (Figure 3, left).

The third experiment addressed a basic prediction of the CoF model, stemming from its reliance on *what+where* mechanisms: the interaction between effects of shape and location in object representation. Such interaction had been found in a psychophysical study [15], in which the task was 4-alternative forced-choice classification of two-part stimuli consisting of simple geometric shapes (cube, cylinder, sphere, cone). The composite stimuli were defined by two variables, shape and location, each of which could be *same, neutral,* or *different* in the prime and the target (yielding 9 conditions altogether). Response times of human subjects revealed effects of shape and location (*what+where*), but not of shape alone; the pattern of priming across the nine conditions was replicated by the CoF model (correlation between model and human data $r = 0.85$), using the same stimuli as in the psychophysical experiment.

# 4 Discussion

Because CoF relies on retinotopy rather than on abstract binding, its representation of spatial structure is location-specific; so is the treatment of structure by the human visual system, as indicated by a number of findings. For example, priming in a subliminal perception task was found to be confined to a quadrant of the visual field [16]. The notion that the representation of an object may be tied to a particular location in the visual field where it is first observed is compatible with the concept of *object file*, a hypothetical record created by the visual system for every encountered object, which persists as long as the object is observed. Moreover, location (as it figures in the CoF model) should be interpreted relative to the focus of attention, rather than retinotopically [17].

The idea that global relationships (hence, large-scale structure) have precedence over local ones [18], which is central to our approach, has withstood extensive testing in the past two decades. Even with the perceptual salience of the global and local structure equated, subjects are able to process the relations among elements before the elements themselves are identified [19]. More generally, humans are limited in their ability to represent spatial structure, in that the representation of spatial relations requires spatial attention. For example, visual search is difficult when

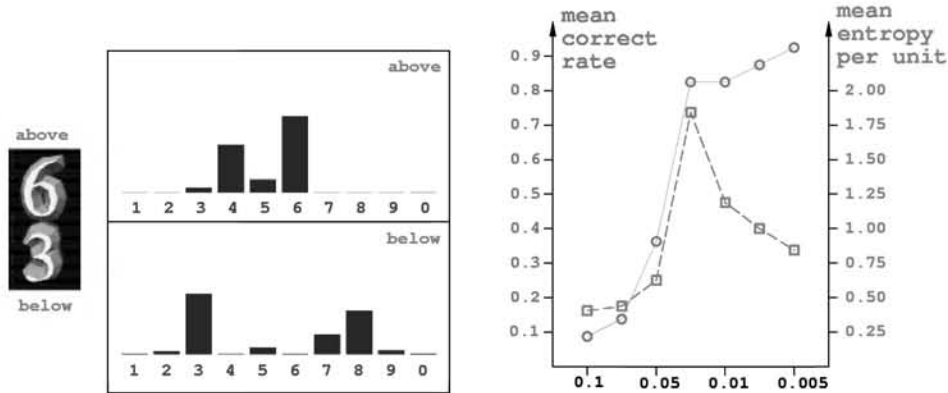

Figure 3: *Left:* the response of the CoF model to a novel composite object, 6 (which only appeared in the bottom position in the training set) over 3 (which was only seen in the top position). The interpretations offered by the model were correct in 94 out of the 100 possible test cases (10 digits on top × 10 digits on the bottom) in this experiment. Note: in the test scenario, each unit (**above** and **below**) must be fed each of the two input fragments (**above** and **below**), hence the 20 bars in the plots of the model's output. *Right:* the non-monotonic dependence of the mean entropy per output unit (ordinate axis on the right; dashed line) on the spread constant $\sigma$ of the radial basis functions (abscissa) indicates that entropy alone should not be used as a training criterion in object representation systems.

targets differ from distractors only in the spatial relation between their elements, as if "...attention is required to bind features ..." [20].

The CoF model offers a unified framework, rooted in the MDL principle, for the understanding of these behavioral findings and of the functional significance of *what+where* receptive fields and attentional gain modulation. It extends the previous use of gain fields in the modeling of translation invariance [21] and of object-centered hemi-neglect [22], and highlights a parallel between *what+where* cells and probabilistic approaches to structure representation in computational vision (e.g., [23]). The representational framework we described is both productive and effectively systematic. Specifically, it has the ability, as a matter of principle, to recognize as such objects that are related through a rearrangement of mesoscopic parts, without being taught those parts individually, and without the need for abstract symbolic binding.

## References

[1] S. Edelman. Computational theories of object recognition. *Trends in Cognitive Science*, 1:296–304, 1997.

[2] J. E. Hummel. Where view-based theories of human object recognition break down: the role of structure in human shape perception. In E. Dietrich and A. Markman, eds., *Cognitive Dynamics: conceptual change in humans and machines*, ch. 7. Erlbaum, Hillsdale, NJ, 2000.

[3] R. F. Hadley. Cognition, systematicity, and nomic necessity. *Mind and Language*, 12:137–153, 1997.

[4] E. Bienenstock, S. Geman, and D. Potter. Compositionality, MDL priors, and object recognition. In M. C. Mozer, M. I. Jordan, and T. Petsche, editors, *NIPS* 9. MIT Press, 1997.

[5] S. Edelman. *Representation and recognition in vision.* MIT Press, Cambridge, MA, 1999.

[6] E. Kobatake and K. Tanaka. Neuronal selectivities to complex object features in the ventral visual pathway of the macaque cerebral cortex. *J. Neurophysiol.*, 71:856–867, 1994.

[7] S. C. Rao, G. Rainer, and E. K. Miller. Integration of what and where in the primate prefrontal cortex. *Science*, 276:821–824, 1997.

[8] C. E. Connor, D. C. Preddie, J. L. Gallant, and D. C. Van Essen. Spatial attention effects in macaque area V4. *J. of Neuroscience*, 17:3201–3214, 1997.

[9] D. J. Heeger, E. P. Simoncelli, and J. Anthony Movshon. Computational models of cortical visual processing. *Proc. Nat. Acad. Sci.*, 93:623–627, 1996.

[10] M. Riesenhuber and T. Poggio. Hierarchical models of object recognition in cortex. *Nature Neuroscience*, 2:1019–1025, 1999.

[11] D. Pomerleau. Input reconstruction reliability estimation. In C. L. Giles, S. J. Hanson, and J. D. Cowan, editors, *NIPS* 5, pages 279–286. Morgan Kaufmann, 1993.

[12] I. Stainvas, N. Intrator, and A. Moshaiov. Improving recognition via reconstruction, 2000. preprint.

[13] S. Edelman and N. Intrator. (Coarse Coding of Shape Fragments) + (Retinotopy) ≈ Representation of Structure. *Spatial Vision*, 13:255–264, 2000.

[14] I. Fujita, K. Tanaka, M. Ito, and K. Cheng. Columns for visual features of objects in monkey inferotemporal cortex. *Nature*, 360:343–346, 1992.

[15] S. Edelman and F. N. Newell. On the representation of object structure in human vision: evidence from differential priming of shape and location. CSRP 500, University of Sussex, 1998.

[16] M. Bar and I. Biederman. Subliminal visual priming. *Psychological Science*, 9(6):464–469, 1998.

[17] A. Treisman. Perceiving and re-perceiving objects. *American Psychologist*, 47:862–875, 1992.

[18] D. Navon. Forest before trees: The precedence of global features in visual perception. *Cognitive Psychology*, 9:353–383, 1977.

[19] B. C. Love, J. N. Rouder, and E. J. Wisniewski. A structural account of global and local processing. *Cognitive Psychology*, 38:291–316, 1999.

[20] A. M. Treisman and N. G. Kanwisher. Perceiving visually presented objects: recognition, awareness, and modularity. *Current Opinion in Neurobiology*, 8:218–226, 1998.

[21] E. Salinas and L. F. Abbott. Invariant visual responses from attentional gain fields. *J. of Neurophysiology*, 77:3267–3272, 1997.

[22] S. Deneve and A. Pouget. Neural basis of object-centered representations. In M. I. Jordan, M. J. Kearns, and S. A. Solla, editors, *NIPS* 11, Cambridge, MA, 1998. MIT Press.

[23] M. C. Burl, M. Weber, and P. Perona. A probabilistic approach to object recognition using local photometry and global geometry. In *Proc. $4^{th}$ Europ. Conf. Comput. Vision, H. Burkhardt and B. Neumann (Eds.), LNCS-Series Vol. 1406–1407, Springer-Verlag*, pages 628–641, June 1998.
